# Approximate inference in continuous time Gaussian-Jump processes

**Manfred Opper**
Fakultät Elektrotechnik und Informatik
Technische Universität Berlin
Berlin, Germany
opperm@cs.tu-berlin.de

**Andreas Ruttor**
Fakultät Elektrotechnik und Informatik
Technische Universität Berlin
Berlin, Germany
andreas.ruttor@tu-berlin.de

**Guido Sanguinetti**
School of Informatics
University of Edinburgh
G.Sanguinetti@ed.ac.uk

## Abstract

We present a novel approach to inference in conditionally Gaussian continuous time stochastic processes, where the latent process is a Markovian jump process. We first consider the case of jump-diffusion processes, where the drift of a linear stochastic differential equation can jump at arbitrary time points. We derive partial differential equations for exact inference and present a very efficient mean field approximation. By introducing a novel lower bound on the free energy, we then generalise our approach to Gaussian processes with arbitrary covariance, such as the non-Markovian RBF covariance. We present results on both simulated and real data, showing that the approach is very accurate in capturing latent dynamics and can be useful in a number of real data modelling tasks.

## Introduction

Continuous time stochastic processes are receiving increasing attention within the statistical machine learning community, as they provide a convenient and physically realistic tool for modelling and inference in a variety of real world problems. Both continuous state space [1, 2] and discrete state space [3–5] systems have been considered, with applications ranging from systems biology [6] to modelling motion capture [7]. Within the machine learning community, Gaussian processes (GPs) [8] have proved particularly popular, due to their appealing properties which allow to reduce the infinite dimensional smoothing problem into a finite dimensional regression problem. While GPs are indubitably a very successful tool in many pattern recognition tasks, their use is restricted to processes with continuously varying temporal behaviour, which can be a limit in many applications which exhibit inherently non-stationary or discontinuous behaviour.

In this contribution, we consider the state inference and parameter estimation problems in a wider class of *conditionally* Gaussian (or Gaussian-Jump) processes, where the mean evolution of the GP is determined by the state of a latent (discrete) variable which evolves according to Markovian dynamics. We first consider the special, but important, case where the GP is a Markovian process, i.e. an Ornstein-Uhlenbeck (OU) process. In this case, exact inference can be derived by using a forward-backward procedure. This leads to partial differential equations, whose numerical solution can be computationally expensive; alternatively, a variational approximation leads to an iterative scheme involving only the numerical solution of ordinary differential equations, and which is extremely efficient from a computational point of view. We then consider the case of general (non-Markov)

GPs coupled to a Markovian latent variable. Inference in this case is intractable, but, by means of a Legendre transform, we can derive a lower bound on the exact free energy, which can be optimised using a saddle point procedure.

# 1 Conditionally Gaussian Markov Processes

We consider a continuous state stochastic system governed by a linear stochastic differential equation (SDE) with piecewise constant (in time) drift bias which can switch randomly with Markovian dynamics (see e.g. [9] for a good introduction to stochastic processes). For simplicity, we give the derivations in the case when there are only two states in the switching process (i.e. it is a random telegraph process) and the diffusion system is one dimensional; generalisation to more dimensions or more latent states is straightforward. The system can be written as

$$dx = (A\mu + b - \lambda x)\, dt + \sigma dw(t),$$
$$\mu(t) \sim \mathcal{TP}\left(f_\pm\right), \tag{1}$$

where $w$ is the Wiener process with variance $\sigma^2$ and $\mu(t)$ is a random telegraph process with switching rates $f_\pm$. Our interest in this type of models is twofold: similar models have found applications in fields like systems biology, where the rapid transitions of regulatory proteins make a switching latent variable a plausible model [6]. At the same time, at least intuitively, model (1) could be considered as an approximation to more complex non-linear diffusion processes, where diffusion near local minima of the potential is approximated by linear diffusion.

Let us assume that we observe the process $x$ at a finite number of time points with i.i.d. noise, giving values

$$y_i \sim \mathcal{N}\left(x(t_i), s^2\right), \qquad i = 1, \ldots, N.$$

For simplicity, we have assumed that the process itself is observed; nothing would change in what follows if we assumed that the variable $y$ is linearly related to the process (except of course that we would have more parameters to estimate). The problem we wish to address is the inference of the joint posterior over both variables $x$ and $\mu$ at *any time* within a certain interval, as well as the determination of (a subset of) the parameters and hyperparameters involved in equation (1) and in the observation model.

## 1.1 Exact state inference

As the system described by equation (1) is a Markovian process, the marginal probability distribution $q_\mu(x, t)$ for both state variables $\mu \in \{0, 1\}$ and $x$ of the posterior process can be calculated using a smoothing algorithm similar to the one described in [6]. Based on the Markov property one can show that

$$q_\mu(x, t) = \frac{1}{Z} p_\mu(x, t)\Psi_\mu(x, t). \tag{2}$$

Here $p_\mu(x, t)$ denotes the marginal filtering distribution, while $\Psi_\mu(x, t) = p(\{y_i | t_i > t\} | x_t = x, \mu_t = \mu)$ is the likelihood of all observations after time $t$ under the condition that the process has state $(x, \mu)$ at time $t$ (backward message). The time evolution of the backward message is described by the backward Chapman-Kolmogorov equation for $\mu \in \{0, 1\}$ [9]:

$$\frac{\partial \Psi_\mu}{\partial t} + (A\mu + b - \lambda x)\frac{\partial \Psi_\mu}{\partial x} + \frac{\sigma^2}{2}\frac{\partial^2 \Psi_\mu}{\partial x^2} = f_{1-\mu}(\Psi_\mu(x, t) - \Psi_{1-\mu}(x, t)). \tag{3}$$

This PDE must be solved backward in time starting at the last observation $y_N$ using the initial condition

$$\Psi_\mu(x, t_N) = p(y_N | x(t_N) = x). \tag{4}$$

The other observations are taken into account by jump conditions

$$\Psi_\mu(x, t_j^-) = \Psi_\mu(x, t_j^+)\, p(y_j | x(t_j) = x), \tag{5}$$

where $\Psi_\mu(x, t_k^\mp)$ being the values of $\Psi_\mu(x, t)$ before and after the $k$-th observation and $p(y_j | x(t_j) = x)$ is given by the noise model.

In order to calculate $q_\mu(x,t)$ we need to calculate the filtering distribution $p_\mu(x,t)$, too. Its time evolution is given by the forward Chapman-Kolmogorov equation [9]

$$\frac{\partial p_\mu}{\partial t} + \frac{\partial}{\partial x}(A\mu + b - \lambda x)p_\mu(x,t) - \frac{\sigma^2}{2}\frac{\partial^2 p_\mu}{\partial x^2} = f_\mu\, p_{1-\mu}(x,t) - f_{1-\mu}\, p_\mu(x,t). \qquad (6)$$

We can show that the posterior process $q_\mu(x,t)$ fulfils a similar PDE by calculating its time derivative and using both (3) and (6). By doing so we find

$$\frac{\partial q_\mu}{\partial t} + \frac{\partial}{\partial x}(A\mu + b - \lambda x + c_\mu(x,t))q_\mu(x,t) - \frac{\sigma^2}{2}\frac{\partial^2 q_\mu}{\partial x^2} = g_\mu(x,t)\, q_{1-\mu}(x,t) - g_{1-\mu}(x,t)\, q_\mu(x,t),$$
$$(7)$$

where

$$g_\mu(x,t) = \frac{\Psi_\mu(x,t)}{\Psi_{1-\mu}(x,t)}\, f_\mu \qquad (8)$$

are time and state dependent posterior jump rates, while the drift

$$c_\mu(x,t) = \sigma^2 \frac{\partial}{\partial x} \log \Psi_\mu(x,t) \qquad (9)$$

takes the observations into account. It is clearly visible that (7) is also a forward Chapman-Kolmogorov equation. Consequently, the only differences between prior and posterior process are the jump rates for the telegraph process $\mu$ and the drift of the diffusion process $x$.

## 1.2 Variational inference

The exact inference approach outlined above gives rise to PDEs which need to be solved numerically in order to estimate the relevant posteriors. For one dimensional GPs this is expensive, but in principle feasible. This work will be deferred to a further publication. Of course, numerical solutions become computationally prohibitive for higher dimensional problems, leading to a need for approximations. We describe here a variational approximation to the joint posterior over the switching process $\mu(t)$ and the diffusion process $x(t)$ which gives an upper bound on the true free energy; it is obtained by making a factorised approximation to the probability over paths $(x_{0:T}, \mu_{0:T})$ of the form

$$q\left(x_{0:T}, \mu_{0:T}\right) = q_x\left(x_{0:T}\right) q_\mu\left(\mu_{0:T}\right), \qquad (10)$$

where $q_x$ is a pure diffusion process (which can be easily shown to be Gaussian) and $q_\mu$ is a pure jump process. Considering the KL divergence between the original process (1) and the approximating process, and keeping into account the conditional structure of the model and equation (10), we obtain the following expression for the Kullback-Leibler (KL) divergence between the true and approximating posteriors:

$$KL\left[q\|p\right] = K_0 + \sum_{i=1}^{N} \langle \log p\left(y_i | x(t_i)\right) \rangle_{q_x} + \langle KL\left[q_x \| p\left(x_{0:T} | \mu_{0:T}\right)\right] \rangle_{q_\mu} + KL\left[q_\mu \| p(\mu_{0:T})\right]. \quad (11)$$

By using the general formula for the KL divergence between two diffusion processes [1], we obtain the following form for the third term in equation (11):

$$\langle KL\left[q_x \| p\left(x_{0:T} | \mu_{0:T}\right)\right] \rangle_{q_\mu} = \int dt\, \frac{1}{2\sigma^2}\{[\alpha(t) + \lambda]^2\left[c^2(t) + m^2(t)\right] + [\beta(t) - b]^2 +$$
$$+ 2\left[\alpha(t) + \lambda\right]\left[\beta(t) - b\right] m(t) + \left[A^2 - 2A\left(\alpha(t) + \lambda\right) m(t) - 2A\left(\beta(t) - b\right)\right] q_\mu^1(t)\}. \qquad (12)$$

Here $\alpha$ and $\beta$ are the gain and bias (coefficients of the linear term and constant) of the drift of the approximating diffusion process, $m$ and $c^2$ are the mean and variance of the approximating process, and $q_\mu^1(t)$ is the marginal probability at time $t$ of the switch being on (computed using the approximating jump process). So the KL is the sum of an initial condition part (which can be set to zero) and two other parts involving the KL between a Markovian Gaussian process and a Markovian Gaussian process observed linearly with noise (second and third terms) and the KL between two telegraph processes. The variational E-step iteratively minimises these two parts using recursions of the forward-backward type. Interleaved with this, variational M-steps can be carried out by optimising the variational free energy w.r.t. the parameters; the fixed point equations for this are easily derived and will be omitted here due to space constraints. Evaluation of the Hessian of the free energy w.r.t. the parameters can be used to provide a measure of the uncertainty associated.

### 1.2.1 Computation of the approximating diffusion process

Minimisation of the second and third term in equation (11) requires finding an approximating Gaussian process. By inspection of equation (12), we see that we are trying to compute the posterior process for a discretely observed Gaussian process with (prior) drift $Aq_\mu^1(t)+b-\lambda x$, with the observations being i.i.d. with Gaussian noise. Due to the Markovian nature of the process, its single time marginals can be computed using the continuous time version of the well known forward-backward algorithm [10, 11]. The single time posterior marginal can be decomposed as

$$q\left(x(t)\right) = p\left(x(t)|y_1,\ldots,y_N\right) = \frac{1}{Z}\phi\left(x(t)\right)\xi\left(x(t)\right), \tag{13}$$

where $\phi$ is the filtered process or forward message, and $\xi$ is the backward message, *i.e.* the likelihood of future observations conditioned on time $t$. The recursions are based on the following general ODEs linking mean $\hat{m}$ and variance $\hat{c}^2$ of a general Gaussian diffusion process with system noise $\sigma^2$ to the drift coefficients $\hat{\alpha}$ and $\hat{\beta}$ of the respective SDE, which are a consequence of the Fokker-Planck equation for Gaussian processes

$$\begin{aligned}
\frac{d\hat{m}}{dt} &= \hat{\alpha}\hat{m} + \hat{\beta}, \\
\frac{d\hat{c}^2}{dt} &= 2\hat{\alpha}\hat{c}^2 + \sigma^2.
\end{aligned} \tag{14}$$

The filtered process outside the observations satisfies the forward Fokker-Planck equation of the prior process, so its mean and variance can be propagated using equations (14) with prior drift coefficients $\hat{\alpha} = -\lambda$ and $\hat{\beta} = Aq_\mu^1 + b$. Observations are incorporated via the jump conditions

$$\lim_{t\to t_i^+}\phi\left(x(t)\right) \propto p\left(y_i|x(t_i)\right)\lim_{t\to t_i^-}\phi\left(x(t)\right), \tag{15}$$

whence the recursions on the mean and variances easily follow. Notice that this is much simpler than (discrete time) Kalman filter recursions as the prior gain is zero in continuous time. Computation of the backward message (smoothing) is analogous; the reader is referred to [10, 11] for further details.

### 1.2.2 Jump process smoothing

Having computed the approximating diffusion process, we now turn to give the updates for the approximating jump process. The KL divergence in equation (11) involves the jump process in two terms: the last term is the KL divergence between the posterior jump process and the prior one, while the third term, which gives the expectation of the KL between the two diffusion processes under the posterior jump, also contains terms involving the jump posterior. The KL divergence between two telegraph processes was calculated in [4]; considering the jump terms coming from equation (12), and adding a Lagrange multiplier to take into account the Master equation fulfilled by the telegraph process, we end up with the following Lagrangian:

$$\mathcal{L}\left[q_\mu, g_\pm, \psi, \xi\right] = KL\left[q_\mu\|p_{\text{prior}}\right] + \int dt \frac{1}{2\sigma^2}\left[A^2 - 2A\left(\alpha + \lambda\right)m - 2A\left(\beta - b\right)\right]q_1(t) +$$
$$\int dt\psi(t)\left(\frac{dq_1}{dt} + (g_- + g_+)q_1 - g_+\right). \tag{16}$$

Notice we use $q_1(t) = q_\mu(\mu(t) = 1)$ to lighten the notation. Functional derivatives w.r.t. to the posterior rates $g_\pm$ allow to eliminate them in favour of the Lagrange multipliers; inserting this into the functional derivatives w.r.t. to the marginals $q_1(t)$ gives ODEs involving the Lagrange multiplier and the prior rates only (as well as terms from the diffusion process), which can be solved backward in time from the condition $\psi(T) = 0$. This allows to update the rates and then the posterior marginals can be found in a forward propagation, in a manner similar to [4].

## 2 Conditionally Gaussian Processes: general case

In this section, we would like to generalise our model to processes of the form

$$dx = (-\lambda x + A\mu + b)dt + df(t), \tag{17}$$

where the white noise driving process $\sigma dw(t)$ in (1) is replaced by an arbitrary GP $df(t)$ [1]. The application of our variational approximation (11) requires the KL divergence $KL[q_x \| p(x_{0:T}|\mu_{0:T})]$ between a GP $q_x$ and a GP with a shifted mean function $p(x_{0:T}|\mu_{0:T})$. Assuming the same covariance this could in principle be computed using the Radon-Nykodym derivative between the two measures. Our preliminary results (based on the Cameron-Martin formula for GPs [12]) indicates that even in simple cases (like Ornstein-Uhlenbeck noise) the measures are not absolutely continuous and the KL divergence is infinite. Hence, we have resorted to a different variational approach, which is based on a lower bound to the free energy.

We use the fact, that conditioned on the path of the switching process $\mu_{0:T}$, the prior of $x(t)$ is a GP with a covariance kernel $K(t, t')$ and can be marginalised out exactly. The kernel $K$ can be easily computed from the kernel of the driving noise process $f(t)$ [2]. In the previous case of white noise $K$ is given by the (nonstationary) Ornstein-Uhlenbeck kernel $K_{OU}(t, t') = \frac{\sigma^2}{2\lambda}\left\{ e^{-\lambda|t-t'|} - e^{-\lambda(t+t')} \right\}$. The mean function of the conditioned GP is obtained by solving the linear ODE (17) without noise, i.e. with $f = 0$. This yields

$$E_{GP}[x(t)|\mu_{0:T}] = \int_0^t e^{-\lambda(t-s)}(A\mu(s) + b) \, ds \, . \tag{18}$$

Marginalising out the conditional GP, the negative log marginal probability of observations (free energy) $F = -\ln p(D)$ is represented as

$$F = -\ln E_\mu\left[p(D|\mu_{0:T})\right] = \kappa - \ln E_\mu\left[\exp\left\{ -\frac{1}{2}(y - x^\mu)^\top(K + \sigma^2 I)^{-1}(y - x^\mu) \right\}\right] \, . \tag{19}$$

Here $E_\mu$ denotes expectation over the prior switching process $p_\mu$, $y$ is the vector of observations, and $x^\mu = E_{GP}[(x(t_1), \ldots, x(t_N))^\top|\mu_{0:T}]$ is the vector of conditional means at observation times $t_i$. $K$ is the kernel matrix and $\kappa = \frac{1}{2}\ln(|2\pi K|)$. This intractable free energy contains a functional in the exponent which is bilinear in the switching process $\mu$. In the spirit of other variational transformations [13, 14] this can be *linearised* through a Legendre transform (or convex duality). Applying $\frac{1}{2}z^\top A^{-1}z = \max_\theta\left\{\theta^\top z - \frac{1}{2}\theta^\top A\theta\right\}$ to the vector $z = (y - x^\mu)$ and the matrix $A = (K + \sigma^2 I)$, and exchanging the $\max$ operation with the expectation over $\mu$, leads to the lower bound

$$F \geq \kappa + \max_\theta\left( -\frac{1}{2}\theta^\top(K + \sigma^2 I)\theta - \ln E_\mu\left[\exp\left\{ -\theta^\top(y - x^\mu) \right\}\right]\right) \, . \tag{20}$$

A similar upper bound which is however harder to evaluate computationally will be presented elsewhere. It can be shown that the lower bound (20) neglects the variance of the $E_\mu[x^\mu]$ process (intuitively, the two point expectations in (19) are dropped). The second term in the bracket looks like the free energy for a jump process model having a (pseudo) log likelihood of the data given by $-\theta^\top(y - x^\mu)$. This auxiliary free energy can again be rewritten in terms of the "standard variational" representation

$$-\ln E_\mu\left[\exp\left\{ -\theta^\top(y - x^\mu) \right\}\right] = \min_q\left\{ KL[q\|p_{\text{prior}}] + \theta^\top(y - E_q[x^\mu]) \right\} \, , \tag{21}$$

where in the second line we have introduced an arbitrary process $q$ over the switching variable and used standard variational manipulations. Inserting (18) into the last term in (21), we see that this KL minimisation is of the same structure as the one in equation (16) with a *linear* functional of $q$ in the (pseudo) likelihood term. Therefore the minimiser $q$ is an inhomogeneous Markov jump process, and we can use a backward and forward sweep to compute marginals $q_1(t)$ *exactly* for a fixed $\theta$! These marginals are used to compute the gradient of the lower bound $(K + \sigma^2 I)\theta + (y - E_q[x^\mu])$ and we iterate between gradient ascent steps and recomputations of $E_q[x^\mu]$. Since the minimax problem defined by (20) and (21) is concave in $\theta$ and convex in $q$ the solution must be unique. Upon convergence, we use the switching process marginals $q_1$ for prediction. Statistics of the smoothed $x$ process can then be computed by summing the conditional GP statistics (obtained by exact GP regression) and the $x^\mu$ statistics, which can be computed using the same methods as in [6].

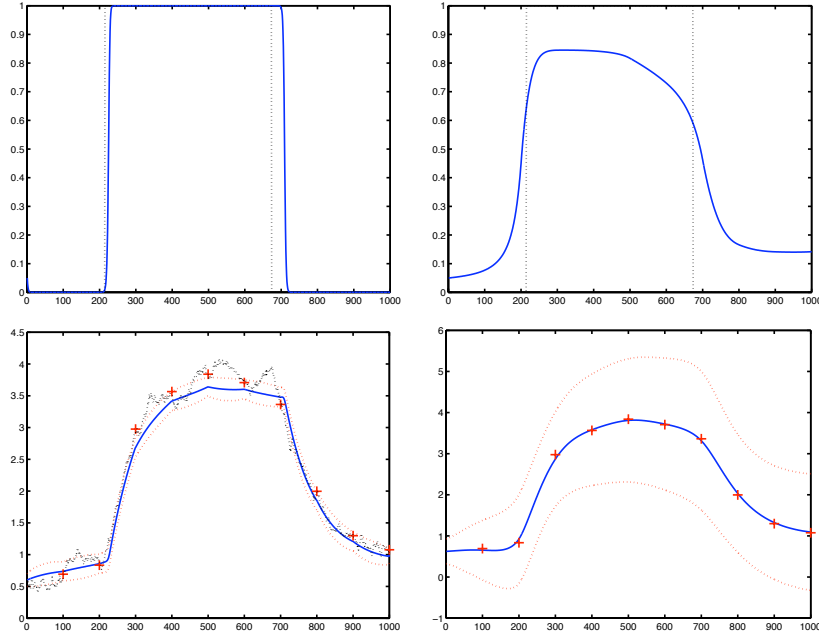

Figure 1: Results on synthetic data. Variational Markovian Gaussian-Jump process on the left, approximate RBF Gaussian-Jump process on the right. Top row, inferred posterior jump means (solid line) and true jump profile (dotted black) Bottom row: inferred posterior mean $x$ (solid) with confidence intervals (dotted red); data points are shown as red crosses, and the true sample profile is shown as black dots. Notice that the less confident jump prediction for the RBF process gives a much higher uncertainty in the $x$ prediction (see text). The $x$ axis units are the simulation time steps.

## 3    Results

### 3.1    Synthetic data

To evaluate the performance and identifiability of our model, we experimented first with a simple one-dimensional synthetic data set generated using a jump profile with only two jumps. A sample from the resulting conditional Gaussian process was then obtained by simulating the SDE using the Euler-Maruyama method, and ten identically spaced points were then taken from the sample path and corrupted with Gaussian noise. Inference was then carried out using two procedures: a Markovian Gaussian-Jump process as described in Section 1, using the variational algorithm, and a "RBF" Gaussian-Jump process with slowly varying covariance, as described in Section 2. The parameters $s^2, \sigma^2$ and $f_\pm$ were kept fixed, while the $A, b$ and $\lambda$ hyperparameters were optimised using type II ML.

The inference results are shown in Figure 1: the left column gives the results of the variational smoothing, while the right column gives the results obtained by fitting a RBF Gaussian-Jump process. The top row shows the inferred posterior mean of the discrete state distribution, while the bottom row gives the conditionally Gaussian posterior. We notice that both approaches provide a good smoothing of the GP and the jump process, although the second jump is inferred as being slightly later than in the true path. Notice that the uncertainties associated with the RBF process are much higher than in the Markovian one, and are dominated by the uncertainty in the posterior mean caused by the uncertainty in the jump process, which is less confident than in the Markovian case (top right figure). This is probably due to the fact that the lower bound (20) ignores the contributions of the variance of the $x^\mu$ term in the free energy, which is due to the variance of the jump process, and hence removes the penalty for having intermediate jump posteriors. A similar behaviour was already noted in a related context in [14]. In terms of computational efficiency, the variational Markovian algorithm converged in approximately 0.1 seconds on a standard laptop, while the RBF process took approximately two minutes. As a baseline, we used a standard discrete time Switching

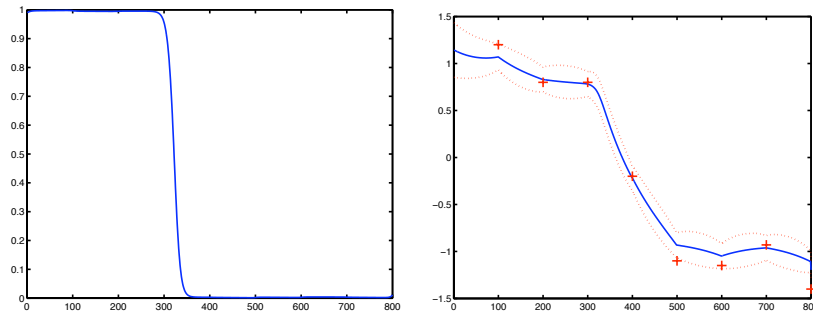

Figure 2: Results on double well diffusion. Left: inferred posterior switch mean; right smoothed data, with confidence intervals. The $x$ axis units are the simulation time steps.

Kalman Filter in the implementation of [15], but did not manage to obtain good results. It is not clear whether the problem resided in the short time series or in our application of the model.

Estimation of the parameters using the variational upper bound also gave very accurate results, with $A = 3.1 \pm 0.3 \times 10^{-2}$ (true value $3 \times 10^{-2}$), $b = 1.0 \pm 2 \times 10^{-2}$ (true value $1 \times 10^{-2}$) and $\lambda = 1.1 \pm 0.1 \times 10^{-2}$ (true value $1 \times 10^{-2}$). It is interesting to note that, if the system noise parameter $\sigma^2$ was set at a higher value, then the $A$ parameter was always driven to zero, leading to a decoupling of the Gaussian and jump processes. In fact, it can be shown that the true free energy has always a local minimum for $A = 0$: heuristically, the GP is always a sufficiently flexible model to fit the data on its own. However, for small levels of system noise, the evidence of the data is such that the more complex model involving a jump process is favoured, giving a type of automated Occam razor, which is one of the main attractions of Bayesian modelling.

## 3.2 Diffusion in a double-well potential

To illustrate the properties of the Gaussian-jump process as an approximator for non-linear stochastic models, we considered the benchmark problem of smoothing data generated from a SDE with double-well potential drift and constant diffusion coefficient. Since the process we wish to approximate is a diffusion process, we use the variational upper bound method, which gave good results in the synthetic experiments. The data we use is the same as the one used in [1], where a non-stationary Gaussian approximation to the non-linear SDE was proposed by means of a variational approximation. The results are shown in Figure 2: as is evident the method both captures accurately the transition time, and provides an excellent smoothing (very similar to the one reported in [1]); these results were obtained in 0.07 seconds, while the Gaussian process approximation of [1] involves gradient descent in a high dimensional space and takes approximately three to four orders of magnitude longer. Naturally, our method cannot be used in this case to estimate the parameters of the true (double well) prior drift, as it only models the linear behaviour near the bottom of each well; however, for smoothing purposes it provides a very accurate and efficient alternative method.

## 3.3 Regulation of competence in *B. subtilis*

Regulation of gene expression at the transcriptional level provides an important application, as well as motivation for the class of models we have been considering. Transcription rates are modulated by the action of transcription factors (TFs), DNA binding proteins which can be activated fast in response to environmental signals. The activation state of a TF is a notoriously difficult quantity to measure experimentally; this has motivated a significant effort within the machine learning and systems biology community to provide models to infer TF activities from more easily measurable gene expression levels [2, 16, 17]. In this section, we apply our model to single cell fluorescence measurements of protein concentrations; the intrinsic stochasticity inherent in single cell data would make conditionally deterministic models such as [2, 6] an inappropriate tool, while our variational SDE model should be able to better capture the inherent fluctuations.

The data we use was obtained in [18] during a study of the genetic regulation of *competence* in *B. subtilis*: briefly, bacteria under food shortage can either enter a dormant stage (spore) or can

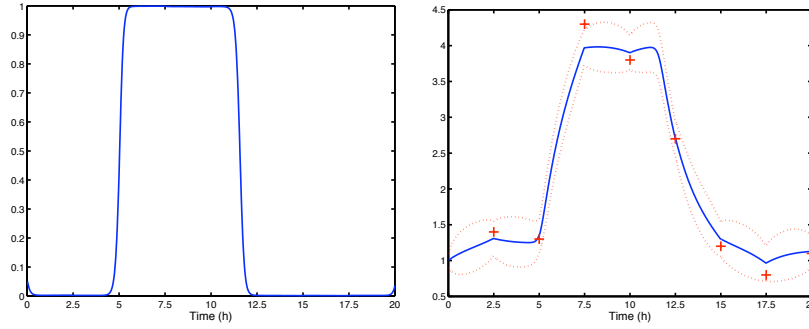

Figure 3: Results on competence circuit. Left: inferred posterior switch mean (ComK activity profile); right smoothed ComS data, with confidence intervals. The $y$ axis units in the right hand panel are arbitrary fluorescence units.

continue to replicate their DNA without dividing (competence). Competence is essentially a bet that the food shortage will be short-lived: in that case, the competent cell can immediately divide into many daughter cells, giving an evolutionary advantage. The molecular mechanisms underpinning competence are quite complex, but the essential behaviour can be captured by a simple system involving only two components: the competence regulator ComK and the auxiliary protein ComS, which is controlled by ComK with a switch-like behaviour (Hill coefficient 5). In [18], ComK activity was indirectly estimated using a gene reporter system (using the ComG promoter). Here, we leave ComK as a latent switching variable, and use our model to smooth the ComS data. The results are shown in Figure 3, showing a clear switch behaviour for ComK activity (as expected, and in agreement with the high Hill coefficient), and a good smoothing of the ComS data. Analysis of the optimal parameters is also instructive: while the $A$ and $b$ parameters are not so informative due to the fact that fluorescence measurements are reported in arbitrary units, the ComS decay rate is estimated as $0.32 \pm 0.06 h^{-1}$, corresponding to a half life of approximately 3 hours, which is clearly plausible from the data. It should be pointed out that, in the simulations in the supplementary material of [18], a nominal value of $0.0014 \ s^{-1}$ was used, corresponding to a half life of only 20 minutes! While the purpose of that simulation was to recreate the qualitative behaviour of the system, rather than to estimate its parameters, the use of such an implausible parameter value illustrates all too well the need for appropriate data-driven tools in modelling complex systems.

## 4   Discussion

In this contribution we proposed a novel inference methodology for continuous time conditionally Gaussian processes. As well as being interesting in its own right as a method for inference in jump-diffusion processes (to our knowledge the first to be proposed), these models find a powerful motivation due to their relevance to fields such as systems biology, as well as plausible approximations to non-linear diffusion processes. We presented both a method based on a variational upper bound in the case of Markovian processes, and a more general lower bound which holds also for non-Markovian Gaussian processes.

A natural question from the machine learning point of view is what are the advantages of continuous time over discrete time approaches. As well as providing a conceptually more correct description of the system, continuous time approaches have at least two significant advantages in our view: a computational advantage in the availability of more stable solvers (such as Runge-Kutta methods), and a communication advantage, as they are more immediately understandable to the large community of modellers which use differential equations but may not be familiar with statistical methods.

There are several possible extension to the work we presented: a relatively simple task would be an extension to a factorial design such as the one proposed for conditionally deterministic systems in [14]. A theoretical task of interest would be a thorough investigation of the relationship between the upper and lower bounds we presented. This is possible, at least for Markovian GPs, but will be presented in other work.

## Footnotes

[1]In case of a process with smooth sample paths, we can write $df(t) = g(t)dt$ with an "ordinary" GP $g$

# References

[1] Cedric Archambeau, Dan Cornford, Manfred Opper, and John Shawe-Taylor. Gaussian process approximations of stochastic differential equations. *Journal of Machine Learning Research Workshop and Conference Proceedings*, 1(1):1–16, 2007.

[2] Neil D. Lawrence, Guido Sanguinetti, and Magnus Rattray. Modelling transcriptional regulation using Gaussian processes. In *Advances in Neural Information Processing Systems 19*, 2006.

[3] Uri Nodelman, Christian R. Shelton, and Daphne Koller. Continuous time Bayesian networks. In *Proceedings of the Eighteenth conference on Uncertainty in Artificial Intelligence (UAI)*, 2002.

[4] Manfred Opper and Guido Sanguinetti. Variational inference for Markov jump processes. In *Advances in Neural Information Processing Systems 20*, 2007.

[5] Ido Cohn, Tal El-Hay, Nir Friedman, and Raz Kupferman. Mean field variational approximation for continuous-time Bayesian networks. In *Proceedings of the twenty-fifthth conference on Uncertainty in Artificial Intelligence (UAI)*, 2009.

[6] Guido Sanguinetti, Andreas Ruttor, Manfred Opper, and Cedric Archambeau. Switching regulatory models of cellular stress response. *Bioinformatics*, 25(10):1280–1286, 2009.

[7] Mauricio Alvarez, David Luengo, and Neil D. Lawrence. Latent force models. In *Proceedings of the Twelfth Interhantional Conference on Artificial Intelligence and Statistics (AISTATS)*, 2009.

[8] Carl E. Rasmussen and Christopher K.I. Williams. *Gaussian Processes for Machine Learning*. MIT press, 2005.

[9] C. W. Gardiner. *Handbook of Stochastic Methods*. Springer, Berlin, second edition, 1996.

[10] Andreas Ruttor and Manfred Opper. Efficient statistical inference for stochastic reaction processes. *Phys. Rev. Lett.*, 103(23), 2009.

[11] Cedric Archambeau and Manfred Opper. Approximate inference for continuous-time Markov processes. In David Barber, Taylan Cemgil, and Silvia Chiappa, editors, *Inference and Learning in Dynamic Models*. Cambridge University Press, 2010.

[12] M. A. Lifshits. *Gaussian Random Functions*. Kluwer, Dordrecht, second edition, 1995.

[13] Michael I. Jordan, Zoubin Ghahramani, Tommi S. Jaakkola, and Lawrence K. Saul. An introduction to variational methods for graphical models. *Machine Learning*, 37:183–233, 1999.

[14] Manfred Opper and Guido Sanguinetti. Learning combinatorial transcriptional dynamics from gene expression data. *Bioinformatics*, 26(13):1623–1629, 2010.

[15] David Barber. Expectation correction for smoothing in switching linear Gaussian state space models. *Journal of Machine Learning Research*, 7:2515–2540, 2006.

[16] James C. Liao, Riccardo Boscolo, Young-Lyeol Yang, Linh My Tran, Chiara Sabatti, and Vwani P. Roychowdhury. Network component analysis: Reconstruction of regulatory signals in biological systems. *Proceedings of the National Academy of Sciences USA*, 100(26):15522–15527, 2003.

[17] Martino Barenco, Daniela Tomescu, David Brewer, Robin Callard, Jaroslav Stark, and Michael Hubank. Ranked prediction of p53 targets using hidden variable dynamical modelling. *Genome Biology*, 7(3), 2006.

[18] Gürol M. Suël, Jordi Garcia-Ojalvo, Louisa M. Liberman, and Michael B. Elowitz. An excitable gene regulatory circuit induces transient cellular differentiation. *Nature*, 440:545–50, 2006.

